# Batch Bayesian Optimization
# via Simulation Matching

**Javad Azimi, Alan Fern, Xiaoli Z. Fern**
School of EECS, Oregon State University
{azimi, afern, xfern}@eecs.oregonstate.edu

## Abstract

Bayesian optimization methods are often used to optimize unknown functions that are costly to evaluate. Typically, these methods sequentially select inputs to be evaluated one at a time based on a posterior over the unknown function that is updated after each evaluation. In many applications, however, it is desirable to perform multiple evaluations in parallel, which requires selecting batches of multiple inputs to evaluate at once. In this paper, we propose a novel approach to batch Bayesian optimization, providing a policy for selecting batches of inputs with the goal of optimizing the function as efficiently as possible. The key idea is to exploit the availability of high-quality and efficient sequential policies, by using Monte-Carlo simulation to select input batches that closely match their expected behavior. Our experimental results on six benchmarks show that the proposed approach significantly outperforms two baselines and can lead to large advantages over a top sequential approach in terms of performance per unit time.

## 1 Introduction

We consider the problem of maximizing an unknown function $f(x)$ when each evaluation of the function has a high cost. In such cases, standard optimization techniques such as empirical gradient methods are not practical due to the high number of function evaluations that they demand. Rather, *Bayesian optimization (BO)* methods [12, 4] have demonstrated significant promise in their ability to effectively optimize a function given only a small number of evaluations. BO gains this efficiency by leveraging Bayesian models that take into account all previously observed evaluations in order to better inform future evaluation choices. In particular, typical BO methods continually maintain a posterior over $f(x)$ that is used to select the next input to evaluate. The result of the evaluation is then used to update the posterior and the process repeats. There are a number of well established policies for selecting the next input to evaluate given the current posterior. We will refer to such policies as *sequential policies* to stress the fact that they select one input at a time.

In many applications it is possible and desirable to run multiple function evaluations in parallel. This is the case, for example, when the underlying function corresponds to a controlled laboratory experiment where multiple experimental setups are examined simultaneously, or when the underlying function is the result of a costly computer simulation and multiple simulations can be run across different processors in parallel. In such cases, existing sequential policies are not sufficient. Rather, batch mode BO is more appropriate, where policies select a batch of multiple inputs to be evaluated at once. To the best of our knowledge and as noted in [4], there is no established work on BO that considers the batch selection problem, except for a brief treatment in [21]. The main contribution of this work is to propose an approach to batch BO and to demonstrate its effectiveness.

The key motivation behind our approach comes from the fact that the sequential mode of BO has a fundamental advantage over BO in batch mode. This is because in sequential mode, each function evaluation is immediately used to obtain a more accurate posterior of $f(x)$, which in turn will allow

a selection policy to make more informed choices about the next input. Given an effective sequential selection policy, our goal is then to design a batch policy that approximates its behavior.

In particular, our batch policy attempts to select a batch that "matches" the expected behavior of a sequential policy as closely as possible. The approach generates Monte-Carlo simulations of a sequential policy given the current posterior, and then derives an optimization problem over possible batches aimed at minimizing the loss between the sequential policy and the batch. We consider two variants of this optimization problem that yield a continuous weighted k-means problem and a combinatorial weighted k-medoid problem. We solve the k-means variant via k-means clustering and show that the k-medoid variant corresponds to minimizing a non-increasing supermodular function, for which there is an efficient approximation algorithm [9].

We evaluate our approach on a collection of six functions and compare it to random and another baseline batch policy based on submodular maximization. The results show that our approach significantly outperforms these baselines and can lead to large advantages over a top sequential approach in terms of performance per unit time.

## 2    Problem Setup

Let $\mathcal{X} \subseteq \mathbb{R}^n$ be an $n$-dimensional input space, where we will often refer to elements of $\mathcal{X}$ as an experiment and assume that each dimension $i$ is bounded in $[A_i, B_i]$. We assume an unknown real-valued function $f : \mathcal{X} \to R$, which represents the expected value of the dependent variable after running an experiment. For example, $f(x)$ might correspond to the result of a wet-lab experiment or a computer simulation with input parameters $x$. Conducting an experiment $x$ produces a noisy outcome $y = f(x) + \epsilon$, where $\epsilon$ is a noise term that might be 0 in some applications.

Our objective is to find an experiment $x \in \mathcal{X}$ that approximately maximizes $f$ by requesting a limited number of experiments and observing their outcomes. Furthermore we are interested in applications where (1) running experiments is costly (e.g. in terms of laboratory or simulation time); and (2) it is desirable to run $k > 1$ experiments in parallel. This motivates the problem of selecting a sequence of batches, each containing $k$ experiments, where the choice of a batch can depend on the results observed from all previous experiments. We will refer to the rule for selecting a batch based on previous experiments as the *batch policy*. The main goal of this paper is to develop a batch policy that optimizes the unknown function as efficiently as possible.

Due to the high cost of experiments, traditional optimization techniques such as empirical gradient ascent are not practical for our setting, due to their high demands on the number of experiments. Rather, we build on Bayesian optimization (BO) [10, 12, 4], which leverages Bayesian modeling in an attempt to achieve more efficient optimization. In particular, BO maintains a posterior over the unknown function based on previously observed experiments, e.g. represented via a Gaussian Process (GP) [19]. This posterior is used to select the next experiment to be run in a way that attempts to trade-off exploring new parts of the experimental space and exploiting parts that look promising. While the BO literature has provided a number of effective policies, they are all *sequential policies*, where only a single experiment is selected and run at a time. Thus, the main novelty of our work is in defining a batch policy in the context of BO, which is described in the next section.

## 3    Simulation Matching for Batch Selection

Given a data set $\mathcal{D}$ of previously observed experiments, which induces a posterior distribution over the unknown function, we now consider how to select the next batch of $k$ experiments. A key issue in making this choice is to manage the trade-off between exploration and exploitation. The policy must attempt to explore by requesting experiments from unexplored parts of the input space, at the same time also attempt to optimize the unknown function via experiments that look promising given the current data. While, under most measures, optimizing this trade-off is computationally intractable, there are a number of heuristic sequential policies from the BO literature that are computationally efficient and perform very well in practice. For example, one such policy selects the next experiment to be the one that has the "maximum expected improvement" according to the current posterior [14, 10]. The main idea behind our approach is to leverage such sequential policies by selecting a batch of $k > 1$ experiments that "closely matches" the sequential policy's expected behavior.

More formally, let $\pi$ be a sequential policy. Given a data set $\mathcal{D}$ of prior experimental results, $\pi$ returns the next experiment $x \in \mathcal{X}$ to be selected. As is standard in BO, we assume we have a posterior

density $P(f \mid \mathcal{D})$ over the unknown function $f$, such as a Gaussian Process. Given this density we can define a density over the outcomes of executing policy $\pi$ for $k$ steps, each outcome consisting of a set of $k$ selected experiments. Let $S_\pi^k$ be the random variable denoting the set of $k$ experiments resulting from such $k$-step executions, which has a well defined density over all possible sets given the posterior of $f$. Importantly, it is generally straightforward to use Monte Carlo simulation to sample values of $S_\pi^k$.[1] Our batch policy is based on generating a number of samples of $S_\pi^k$, which are used to define an objective for optimizing a batch of $k$ experiments. Below we describe this objective and a variant, followed by a description of how we optimize the proposed objectives.

## 3.1 Batch Objective Function

Our goal is to select a batch $B$ of $k$ experiments that best "matches the expected behavior" of a base sequential policy $\pi$ conditioned on the observed data $\mathcal{D}$. More precisely, we consider a batch $B$ to be a good match for a policy execution if $B$ contains an experiment that is close to the best of the $k$ experiments selected by the policy. To specify this objective we first introduce some notation. Given a function $f$ and a set of experiments $S$, we define $x^*(f, S) = \arg\max_{x \in S} f(x)$ to be the maximizer of $f$ in $S$. Also, for any experiment $x$ and set $B$ we define $\mathrm{nn}(x, B) = \arg\min_{x' \in B} \| x - x' \|$ to be the nearest neighbor of $x$ in set $B$. Our objective can now be written as selecting a batch $B$ that minimizes

$$\mathrm{OBJ}(B) = \mathbb{E}_{S_\pi^k} \left[ \mathbb{E}_{f \mid S_\pi^k} \left[ \| x^*(f, S_\pi^k) - \mathrm{nn}(x^*(f, S_\pi^k), B) \|^2 \mid \mathcal{D} \right] \mid \mathcal{D} \right].$$

Note that this nested expectation is the result of decomposing the joint posterior over $S_\pi^k$ and $f$ as $P(f, S_\pi^k \mid \mathcal{D}) = P(f \mid S_\pi^k, \mathcal{D}) \cdot P(S_\pi^k \mid \mathcal{D})$. If we assume that the unknown function $f(x)$ is *Lipschitz continuous* then minimizing this objective can be viewed as minimizing an upper bound on the expected performance difference between the sequential policy and the selected batch. Here the performance of a policy or a batch is equal to the output value of the best selected experiment.

We will approximate this objective by replacing the outer expectation over $S_\pi^k$ with a sample average over $n$ samples $\{S_1, \ldots, S_n\}$ of $S_\pi^k$ as follows, recalling that each $S_i$ is a set of $k$ experiments:

$$
\begin{aligned}
\mathrm{OBJ}(B) &\approx \frac{1}{n} \sum_i \mathbb{E}_{f \mid S_i} \left[ \| x^*(f, S_i) - \mathrm{nn}(x^*(f, S_i), B) \|^2 \mid \mathcal{D} \right] \\
&= \frac{1}{n} \sum_i \sum_{x \in S_i} \Pr(x = x^*(f, S_i) \mid \mathcal{D}, S_i) \cdot \| x - \mathrm{nn}(x, B) \|^2 \\
&= \frac{1}{n} \sum_i \sum_{x \in S_i} \alpha_{i,x} \cdot \| x - \mathrm{nn}(x, B) \|^2 \quad\quad (1)
\end{aligned}
$$

The second step follows by noting that $x^*(f, S_i)$ must be one of the $k$ experiments in $S_i$.

We now define our objective as minimizing (1) over batch $B$. The objective corresponds to a weighted k-means clustering problem, where we must select $B$ to minimize the weighted distortion between the simulated points and their closest points in $B$. The weight on each simulated experiment $\alpha_{i,x}$ corresponds to the probability that the experiment $x \in S_i$ achieves the maximum value of the unknown $f$ among the experiments in $S_i$, conditioned on $\mathcal{D}$ and the fact that $S_\pi^k = S_i$. We refer to this objective as the *k-means objective*.

We also consider a variant of this objective where the goal is to find a $B$ that minimizes (1) under the constraint that $B$ is restricted to experiments in the simulations, i.e. $B \subseteq \bigcup_i S_i$ s.t. $|B| = k$. This objective corresponds to the weighted $k$-medoid clustering problem, which is often considered to improve robustness to outliers in clustering. Accordingly we will refer to this objective as the *k-medoid objective* and note that given a fixed set of simulations this corresponds to a discrete optimization problem.

## 3.2 Optimization Approach

The above $k$-means and $k$-medoid objectives involve the weights $\alpha_{i,x} = P(x = x_i^*(f) \mid \mathcal{D}, S_\pi^k = S_i)$, for each $x \in S_i$. In general these weights will be difficult to compute exactly, particularly

**Algorithm 1** Greedy Weighted $k$-Medoid Algorithm

---
**Input:**$\mathcal{S} = \{(x_1, w_1), \ldots, (x_m, w_m)\}, k$
**Output:**$B$
  $B \leftarrow \{x_1, \ldots, x_m\}$ // initialize batch to all data points
  **while** $|B| > k$ **do**
    $x \leftarrow \arg\min_{x \in B} \sum_{j=1}^m w_j \cdot \| x_j - \mathrm{nn}(x_j, B \setminus x) \|$ // point that influences objective the least
    $B \leftarrow B \setminus x$
  **end while**
  return $B$

---

due to the conditioning on the set $S_i$. In this work, we approximate those weights by dropping the conditioning on $S_i$, for which it is then possible to derive a closed form when the posterior over $f$ is represented as a Gaussian Process (GP). We have found that this approach leads to good empirical performance. In particular, instead of using the weights $\alpha_{i,x}$ we use the weights $\hat{\alpha}_{i,x} = P(x = x_i^*(f) \mid \mathcal{D})$. When the posterior over $f$ is represented as a GP, as in our experiments, the joint distribution over experimental outcomes in $S_i = \{x_{i,1}, \ldots, x_{i,k}\}$ is normally distributed. That is, the random vector $\langle f(x_{i,1}), \ldots, f(x_{i,k}) \rangle \sim \mathcal{N}(\boldsymbol{\mu}, \boldsymbol{\Sigma})$, where the mean $\boldsymbol{\mu}$ and covariance $\boldsymbol{\Sigma}$ have standard closed forms given by the GP conditioned on $\mathcal{D}$. From this, it is clear that for a GP the computation of $\hat{\alpha}_{i,x}$ is equivalent to computing the probability that the $i^{th}$ component of a normally distributed vector is larger than the other components. A closed form solution for this probability is given by the following proposition.

**Proposition 1.** *If $(y_1, y_2, \ldots, y_k) \sim \mathcal{N}(\boldsymbol{\mu_y}, \boldsymbol{\Sigma_y})$ then for any $i \in \{1, \ldots, k\}$,*

$$P(y_i \geq y_1, y_i \geq y_2, \ldots, y_i \geq y_k) = \prod_{j=1}^{k-1} (1 - \Phi(-\mu_j)) \tag{2}$$

*where $\Phi(.)$ is standard normal cdf, $\boldsymbol{\mu} = (\mu_1, \mu_2, \cdots, \mu_{k-1}) = \left(A\boldsymbol{\Sigma_y}A'\right)^{-\frac{1}{2}} A\boldsymbol{\mu_y}$, such that $A \in \mathbb{R}^{(k-1) \times k}$ is a sparse matrix that for any $j = 1, 2, \cdots, k-1$ we have $A_{j,i} = 1$, and for any $1 \leq p < i$ we have $A_{p,p} = -1$, and for any $i < p \leq k$ we have $A_{p-1,p} = -1$.*

Using this approach to compute the weights we can now consider optimizing the $k$-means and $k$-medoid objectives from (1), both of which are known to be NP-hard problems. For the $k$-means objective we solve for the set $B$ by simply applying the $k$-means clustering algorithm [13] to the weighted data set $\bigcup_i \bigcup_{x \in S_i} \{(x, \hat{\alpha}_{i,x})\}$. The $k$ cluster centers are returned as our batch $B$.

The $k$-medoid objective is well known [22] and the weighted k-medoid clustering algorithm [11] has been shown to perform well and be robust to outliers in the data. While we have experimented with this algorithm and obtained good results, we have achieved results that are as good or better using an alternative greedy algorithm that provides certain approximation guarantees. Pseudo-code for this algorithm is shown in Figure 1. The input to the algorithm is the set of weighted experiments and the batch size $k$. The algorithm initializes the batch $B$ to include all of the input experiments, which achieves the minimum objective value of zero. The algorithm then iteratively removes one experiment from $B$ at a time until $|B| = k$, each time removing the element whose removal results in the smallest increase in the $k$-medoid objective.

This greedy algorithm is motivated by theoretical results on the minimization of non-increasing, supermodular set functions.

**Definition 1.** *Suppose $\mathcal{S}$ is a finite set, $f : 2^{\mathcal{S}} \rightarrow \mathbb{R}^+$ is a supermodular set function if for all $B_1 \subseteq B_2 \subseteq \mathcal{S}$ and $\{x\} \in \mathcal{S} \setminus B_2$, it holds that $f(B_1) - f(B_1 \cup \{x\}) \geq f(B_2) - f(B_2 \cup \{x\})$.*

Thus, a set function is supermodular if adding an element to a smaller set provides no less improvement than adding the element to a larger set. Also, a set function is non-increasing if for any set $S$ and element $x$ if $f(S) \geq f(S \cup \{x\})$. It can be shown that our $k$-medoid objective function of (1) is both a non-increasing and supermodular function of $B$ and achieves a minimum value of zero for $B = \bigcup_i S_i$. It follows that we can obtain an approximation guarantee for the described greedy algorithm in [9].

**Theorem 1.** *[9] Let $f$ be a monotonic non-increasing supermodular function over subsets of the finite set $\mathcal{S}$, $|\mathcal{S}| = m$ and $f(\mathcal{S}) = 0$. Let $B$ be the set of the elements returned by the greedy algorithm 1 s.t $|B| = k$, $q = m - k$ and $B^* = \operatorname{argmin}_{B' \subseteq \mathcal{S}, |B'| = k} f(B')$, then*

$$f(B) \leq \frac{1}{t} \left[ \left( \frac{q+t}{q} \right)^q - 1 \right] f(B^*) \leq \frac{e^t - 1}{t} f(B^*) \tag{3}$$

*where $t$ is the steepness parameter [9] of function $f$.*

Notice that the approximation bound involves the steepness parameter $t$ of $f$, which characterizes the rate of decrease of $f$. This is unavoidable since it is known that achieving a constant factor approximation guarantee is not possible unless P=NP [17]. Further this bound has been shown to be tight for any $t$ [9]. Note that this is in contrast to guarantees for greedy maximization of submodular functions [7] for which there are constant factor guarantees. Also note that the greedy algorithm we use is qualitatively different from the one used for submodular maximization, since it greedily removes elements from $B$ rather than greedily adding elements to $B$.

## 4 Implementation Details and Baselines

**GP Posterior.** Our batch selection approach described above requires that we maintain a posterior over the unknown function $f$. For this purpose we use a zero-mean GP prior with a zero-mean Gaussian noise model with variance equal to 0.01. The GP covariance is specified by a Gaussian kernel $K(x, x') = \sigma \exp \left( -\frac{1}{2w} \| x - x' \|^2 \right)$, with signal variance $\sigma = y_{\max}^2$ where $y_{\max}$ is the maximum value of the unknown function. In all of our experiments we used a simple rule of thumb to set the kernel width $w$ to $0.01 \sum_{i=1}^d l_i$ where $l_i$ is the input space length in dimension $i$. We have found this rule to work well for a variety of problems. An alternative would be to use a validation-based approach for selecting the kernel parameters. In the BO setting, however, we have found this to be unreliable since the number of data points is relatively small.

**Base Sequential Policy.** Our batch selection approach also requires a base sequential policy $\pi$ to be used for simulation matching. This policy must be able to select the next experiment given any set of prior experimental observations $\mathcal{D}$. In our experiments, we use a policy based on the *Maximum Expected Improvement* (MEI) heuristic [14, 10] which is a very successful sequential policy for BO and has been shown to converge in the limit to the global optimum. Given data $\mathcal{D}$ the MEI policy simply selects the next experiment to be the one that maximizes the expected improvement over the current set of experiments with respect to maximizing the unknown function. More formally, let $y^*$ be the value of the best/largest experimental outcome observed so far in $\mathcal{D}$. The MEI value of an experiment $x$ is given by $\text{MEI}(x) = \mathbb{E}_f \left[ \max\{ f(x) - y^*, 0 \} \mid \mathcal{D} \right]$. For our GP posterior over $f$ we can derive a closed form for this given by: $u = \frac{y^* - \mu(x)}{\sigma(x)}$ where $y^*$ is our best currently observed value. For any given example $x$, the MEI can be computed as follows:

$$\text{MEI}(x) \quad = \quad \sigma(x) \left[ -u\Phi(-u) + \phi(u) \right], \quad u = \frac{y^* - \mu(x)}{\sigma(x)}$$

where $\Phi$ and $\phi$ are the standard normal cumulative distribution and density functions and $\mu(x)$ and $\sigma(x)$ are the mean and variance of $f(x)$ according to the GP given $\mathcal{D}$, which have simple closed forms. Note that we have also evaluated our simulation-matching approach with an alternative sequential policy known as *Maximum Probability of Improvement* [16, 10]. The results (not shown in this paper) are similar to those obtained from MEI, showing that our general approach works well for different base policies.

The computation of the MEI policy requires maximizing $\text{MEI}(x)$ over the input space $\mathcal{X}$. In general, this function does not have a unique local maximum and various strategies have been tried for maximizing it. In our experiments, we (approximately) maximize the MEI function using the *DIRECT* black-box optimization procedure, which has shown good optimization performance as well as computational efficiency in practice.

**Baseline Batch Policies.** To the best of our knowledge there is no *well-known* batch policy for Bayesian optimization. However, in our experiments we will compare against two baselines. The first baseline is random selection, where a batch of $k$ random experiments is returned at each step. Interestingly, in the case of batch active learning for classification, the random batch selection strategy

Table 1: Benchmark Functions.

| Function | Mathematical representation |
|---|---|
| Cosines | $1 - (u^2 + v^2 - 0.3cos(3\pi u) - 0.3cos(3\pi v))$    $u = 1.6x - 0.5, v = 1.6y - 0.5$ |
| Rosenbrock | $10 - 100(y - x^2)^2 - (1 - x)^2$ |
| Michalewicz | $-\sum_{i=1}^{5} \sin(x_i) . \left( \sin\left( \frac{i.x_i^2}{\pi} \right) \right)^{20}$ |

has been surprisingly effective and is often difficult to outperform with more sophisticated strategies [8]. However, as our experiments will show, our approach will dominate random.

Our second, more sophisticate, baseline is based on selecting a batch of experiments whose expected maximum output is the largest. More formally, we consider selecting a size $k$ batch $B$ that maximizes the objective $\mathbb{E}_f [\max_{x \in B} f(x) \mid \mathcal{D}]$, which we will refer to as the EMAX objective. For our GP prior, each set $B = \{x_1, \ldots, x_k\}$ can be viewed as defining a normally distributed vector $\langle f(x_1), \ldots, f(x_k) \rangle \sim \mathcal{N}(\boldsymbol{\mu}, \boldsymbol{\Sigma})$. Even in this case, finding the optimal set $B$ is known to be NP-hard. However, for the case where $f$ is assumed to be non-negative, the EMAX objective is a non-negative, submodular, non-decreasing function of $B$. Together these properties imply that a simple greedy algorithm can achieve an approximation ratio of $1 - e^{-1}$ [7]. The algorithm starts with an empty $B$ and greedily adds experiments to $B$, each time selecting the one that improves the EMAX objective the most. Unfortunately, in general there is no closed form solution for evaluating the EMAX objective, even in our case of normally distributed vectors [20]. Therefore, to implement the greedy algorithm, which requires many evaluations of the EMAX objective, we use Monte Carlo sampling, where for a given set $B$ we sample the corresponding normally distributed vector and average the maximum values across the samples.

## 5 Experimental Results

In this section we evaluate our proposed batch BO approach and the baseline approaches on six different benchmarks.

### 5.1 Benchmark Functions

We consider three well-known synthetic benchmark functions: *Cosines* and *Rosenbrock* [1, 5], which are over $[0, 1]^2$, and *Michalewicz* [15], which is over $[0, \pi]^5$. Table 1 gives the formulas for each of these functions. Two additional benchmark functions *Hydrogen* and *FuelCell*, which range over $[0, 1]^2$, are derived from real-world experimental data sets. In both cases, the benchmark function was created by fitting regression models to data sets resulting from real experiments. The Hydrogen data set is the result of data collected as part of a study on biosolar hydrogen production [6], where the goal was to maximize the hydrogen production of a particular bacteria by optimizing the PH and Nitrogen levels of the growth medium. The FuelCell data set was collected as part of a study investigating the influence of anodes' nano-structure on the power output of microbial fuel cells [3]. The experimental inputs include the average area and average circularity of the nano-particles [18]. Contour plots of the four 2-$d$ functions are shown in Figure 1.

The last benchmark function is derived from the Cart-Pole [2] problem, which is a commonly used reinforcement learning problem. The goal is to optimize the parameters of a controller for a wheeled cart with the objective of balancing a pole. The controller is parameterized by four parameters giving a 4-$d$ space of experiments in $[1, -1]^4$. Given a setting for these parameters, the benchmark function is implemented by using the standard Cart-Pole simulator to return the reward received for the controller.

### 5.2 Results

Figures 2 and 3 show the performance of our methods on all six benchmark functions for batch sizes 5 and 10 respectively. Each graph contains 5 curves, each corresponding to a different BO approach (see below). Each curve is the result of taking an average of 100 independent runs. The $x$-axis of each graph represents the total number of experiments and the $y$-axis represents the regret values, where the regret of a policy at a particular point is the difference between the best possible output value (or an upper bound if the value is not known) and the best value found by the policy. Hence the regret is always positive and smaller values are preferred. Each run of a policy initializes the data set to contain 5 randomly selected experiments for the 2-$d$ functions and 20 random initial experiments for the higher dimensional functions.

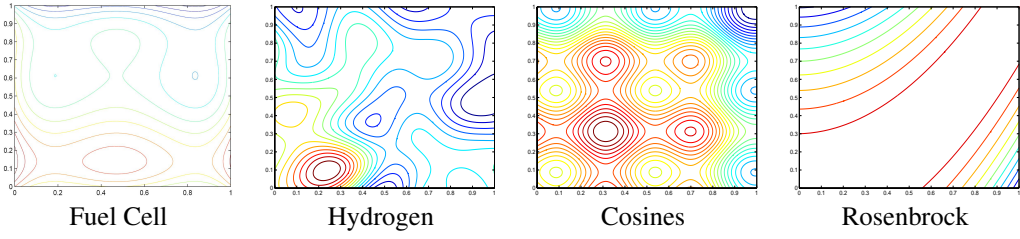

Figure 1: The contour plots for the four 2−dimension proposed test functions.

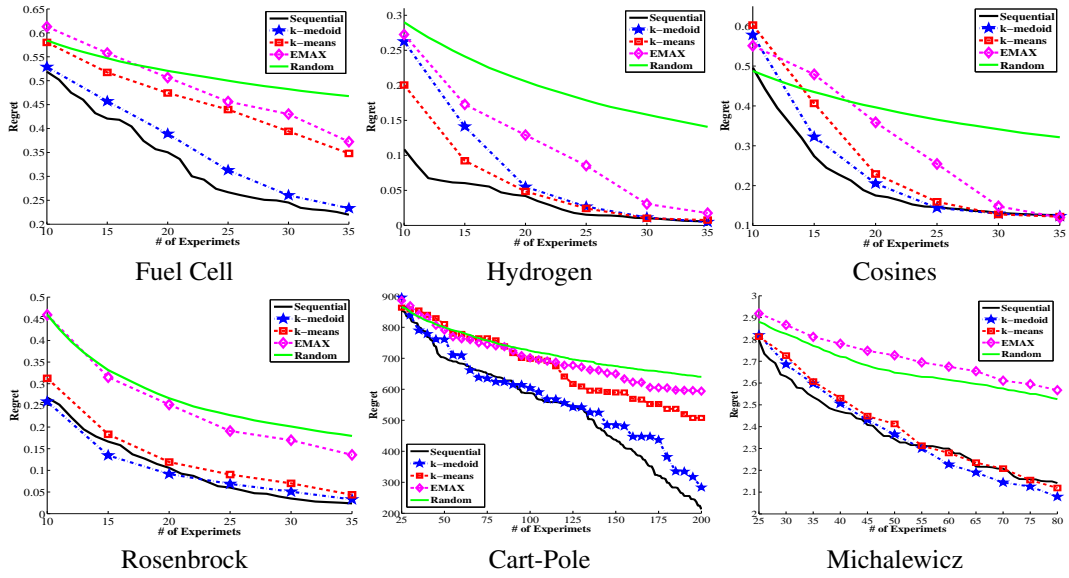

Figure 2: Performance evaluation with batch size 5.

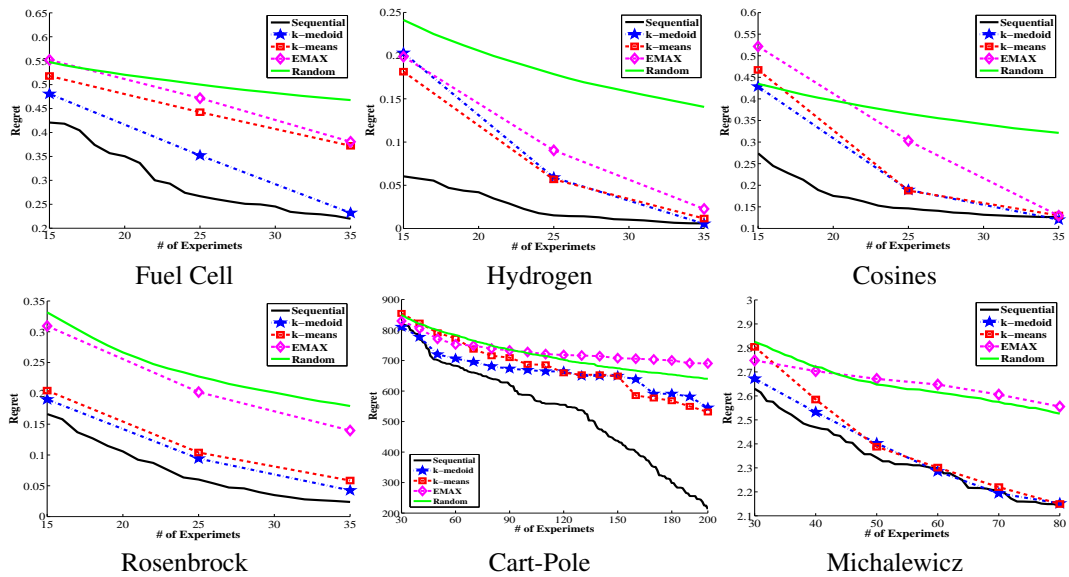

Figure 3: Performance evaluation with batch size 10.

Each graph gives curves for four batch approaches including our baselines *Random* and *EMAX*, along with our proposed approaches based on the $k$-*means* and $k$-*medoid* objectives, which are optimized by weighted $k$-means clustering and the greedy Algorithm 1 respectively. In addition, for reference we plot the performance of the base *Sequential* MEI BO policy ($k = 1$) on each graph. Note that since the batch approaches request either 5 or 10 experiments at a time, their curves only contain data points at those intervals. For example, for the batch size 5 results the first point on a batch curve corresponds to 10 experiments, including the initial 5 experiments and the first requested batch. The next point on the batch curve is for 15 experiments which includes the next requested batch and so on. Rather the Sequential policy has a point at every step since it requests experiments one at a time. It is important to realize that we generally expect a good sequential policy to do better, or no worse, than a batch policy with respect to performance per number of experiments. Thus, the Sequential curve can be typically viewed as an upper performance bound and provides an indication of how much loss is incurred when moving to a batch setting in terms of efficiency per experiment.

**Comparison to Baselines.** The major observation from our results is that for all benchmarks and for both batch sizes the proposed $k$-means and $k$-medoid approaches significantly outperform the baselines. This provides strong validation for our proposed simulation-matching approach to batch selection.

$k$**-means vs.** $k$**-medoid.** In most cases, the $k$-means and $k$-medoid approaches perform similarly. However, for both batch sizes $k$-medoid often does shows a small improvement over $k$-means and appears to have a significant advantage in FuelCell. The only exception is in Hydrogen where $k$-means shows a small advantage over $k$-medoid for small numbers of experiments. Overall, both approaches appear to be effective and in these domains $k$-medoid has a slight edge.

**Batch vs. Sequential.** The advantage of Sequential over our batch approaches varies with the benchmark. However, in most cases, our proposed batch approaches catch up to Sequential in a relatively small number of experiments and in some cases, the batch policies are similar to Sequential from the start. The main exception is Cart-Pole for batch size 10, where the batch policies appear to be significantly less efficient in terms of performance versus number of experiments. Generally, we see that the difference between our batch policies and Sequential is larger for batch size 10 than batch size 5, which is expected, since larger batch sizes imply that less information per experiment is used in making decisions.

It is clear, however, that if we evaluate the performance of our batch policies in terms of experimental time, then there is a very significant advantage over Sequential. In particular, the amount of experimental time for a policy is approximately equal to the number of requested batches, assuming that the batch size is selected to allow for all selected experiments to be run in parallel. This means, for example, that for the batch size 5 results, 5 time steps for the batch approaches correspond to 30 total experiments (5 initial + 5 batches). We can compare this point to the first point on the Sequential curve, which also corresponds to 5 time steps (5 experiments beyond the initial 5). In all cases, the batch policies yield a very large improvement in regret reduction per unit time, which is the primary motivation for batch selection.

## 6 Summary and Future Work

In this paper we introduced a novel approach to batch BO based on the idea of simulation matching. The key idea of our approach is to design batches of experiments that approximately match the expected performance of high-quality sequential policies for BO. We considered two variants of the matching problem and showed that both approaches significantly outperformed two baselines including random batch selection on six benchmark functions. For future work we plan to consider the general idea of simulation matching for other problems, such as active learning, where there are also good sequential policies and batch selection is often warranted. In addition, we plan to consider less myopic approaches for selecting each batch and the problem of batch size selection, where there is a choice about batch size that must take into account the current data and experimental budget.

## Acknowledgments

The authors acknowledge the support of the NSF under grants IIS-0905678.

## Footnotes

[1]For example, this can be done by starting with $\mathcal{D}$ and selecting the first experiment $x_1$ using $\pi$ and then using $P(f \mid \mathcal{D})$ to simulate the result $y_1$ of experiment $x_1$. This simulated experiment is added to $\mathcal{D}$ and the process repeats for $k - 1$ additional experiments.

# References

[1] B. S. Anderson, A. W. More, and D. Cohn. A nonparametric approach to noisy and costly optimization. In *ICML*, 2000.

[2] A. G. Barto, R. S. Sutton, and C. W. Anderson. Neuronlike adaptive elements that can solve difficult learning control problems. 13:835–846, 1983.

[3] D. Bond and D. Lovley. Electricity production by geobacter sulfurreducens attached to electrodes. *Applications of Environmental Microbiology*, 69:1548–1555, 2003.

[4] E. Brochu, M. Cora, and N. de Freitas. A tutorial on Bayesian optimization of expensive cost functions, with application to active user modeling and hierarchical reinforcement learning. Technical Report TR-2009-23, Department of Computer Science, University of British Columbia, 2009.

[5] M. Brunato, R. Battiti, and S. Pasupuleti. A memory-based rash optimizer. In *AAAI-06 Workshop on Heuristic Search, Memory Based Heuristics and Their applications*, 2006.

[6] E. H. Burrows, W.-K. Wong, X. Fern, F. W. Chaplen, and R. L. Ely. Optimization of ph and nitrogen for enhanced hydrogen production by synechocystis sp. pcc 6803 via statistical and machine learning methods. *Biotechnology Progress*, 25:1009–1017, 2009.

[7] M. F. G Nemhauser, L Wolsey. An analysis of the approximations for maximizing submodular set functions. *Mathematical Programmingn*, 14:265–294, 1978.

[8] Y. Guo and D. Schuurmans. Discriminative batch mode active learning. *Proceedings of Advances in Neural Information Processing Systems (NIPS2007)*, 6, 2007.

[9] V. P. Il'ev. An approximation guarantee of the greedy descent algorithm for minimizing a supermodular set function. *Discrete Applied Mathematics*, 114(1-3):131–146, 2001.

[10] D. Jones. A taxonomy of global optimization methods based on response surfaces. *Journal of Global Optimization*, 21:345–383, 2001.

[11] L. Kaufman and P. J. Rousseeuw. Clustering by means of medoids. *Statistical data analysis based on L1 norm*, pages 405–416, 1987.

[12] D. Lizotte. *Practical Bayesian optimization*. PhD thesis, University of Alberta, 2008.

[13] S. Lloyd. Least squares quantization in PCM. *IEEE Transactions on Information Theory*, 28(2):129–137, 1982.

[14] M. Locatelli. Bayesian algorithms for one-dimensional globaloptimization. *J. of Global Optimization*, 10(1):57–76, 1997.

[15] Z. Michalewicz. *Genetic algorithms + data structures = evolution programs (2nd, extended ed.)*. Springer-Verlag New York, Inc., New York, NY, USA, 1994.

[16] A. Moore and J. Schneider. Memory-based stochastic optimization. In *NIPS*, 1995.

[17] G. Nemhauser and L. Wolsey. *Integer and combinatorial optimization*. Wiley New York, 1999.

[18] D. Park and J. Zeikus. Improved fuel cell and electrode designs for producing electricity from microbial degradation. *Biotechnol.Bioeng.*, 81(3):348–355, 2003.

[19] C. E. Rasmussen and C. K. I. Williams. *Gaussian Processes for Machine Learning*. MIT, 2006.

[20] A. M. Ross. Computing Bounds on the Expected Maximum of Correlated Normal Variables . *Methodology and Computing in Applied Probability*, 2008.

[21] M. Schonlau. *Computer Experiments and Global Optimization*. PhD thesis, University of Waterloo, 1997.

[22] H. D. Vinod. Integer programming and the theory of grouping. *Journal of the American Statistical Association*, 64(326):506–519, 1969.

